# 1-norm Support Vector Machines

**Ji Zhu, Saharon Rosset, Trevor Hastie, Rob Tibshirani**
Department of Statistics
Stanford University
Stanford, CA 94305
{jzhu,saharon,hastie,tibs}@stat.stanford.edu

## Abstract

The standard 2-norm SVM is known for its good performance in two-class classification. In this paper, we consider the 1-norm SVM. We argue that the 1-norm SVM may have some advantage over the standard 2-norm SVM, especially when there are redundant noise features. We also propose an efficient algorithm that computes the whole solution path of the 1-norm SVM, hence facilitates adaptive selection of the tuning parameter for the 1-norm SVM.

## 1 Introduction

In standard two-class classification problems, we are given a set of training data $(x_1, y_1)$, $\dots (x_n, y_n)$, where the input $x_i \in \mathcal{R}^p$, and the output $y_i \in \{1, -1\}$ is binary. We wish to find a classification rule from the training data, so that when given a new input $x$, we can assign a class $y$ from $\{1, -1\}$ to it.

To handle this problem, we consider the 1-norm support vector machine (SVM):

$$\min_{\beta_0,\beta} \quad \sum_{i=1}^{n}\left[1 - y_i\left(\beta_0 + \sum_{j=1}^{q}\beta_j h_j(x_i)\right)\right]_+ \tag{1}$$

$$\text{s.t.} \quad \|\beta\|_1 = |\beta_1| + \cdots + |\beta_q| \leq s, \tag{2}$$

where $\mathcal{D} = \{h_1(x), \dots h_q(x)\}$ is a dictionary of basis functions, and $s$ is a tuning parameter. The solution is denoted as $\hat{\beta}_0(s)$ and $\hat{\beta}(s)$; the fitted model is

$$\hat{f}(x) = \hat{\beta}_0 + \sum_{j=1}^{q}\hat{\beta}_j h_j(x). \tag{3}$$

The classification rule is given by $sign[\hat{f}(x)]$. The 1-norm SVM has been successfully used in [1] and [9]. We argue in this paper that the 1-norm SVM may have some advantage over the standard 2-norm SVM, especially when there are redundant noise features.

To get a good fitted model $\hat{f}(x)$ that performs well on future data, we also need to select an appropriate tuning parameter $s$. In practice, people usually pre-specify a finite set of values for $s$ that covers a wide range, then either use a separate validation data set or use

cross-validation to select a value for $s$ that gives the best performance among the given set. In this paper, we illustrate that the solution path $\hat{\beta}(s)$ is piece-wise linear as a function of $s$ (in the $\mathcal{R}^q$ space); we also propose an efficient algorithm to compute the exact whole solution path $\{\hat{\beta}(s), 0 \leq s \leq \infty\}$, hence help us understand how the solution changes with $s$ and facilitate the adaptive selection of the tuning parameter $s$. Under some mild assumptions, we show that the computational cost to compute the whole solution path $\hat{\beta}(s)$ is $O(nq\min(n,q)^2)$ in the worst case and $O(nq)$ in the best case.

Before delving into the technical details, we illustrate the concept of piece-wise linearity of the solution path $\hat{\beta}(s)$ with a simple example. We generate 10 training data in each of two classes. The first class has two standard normal independent inputs $x_1, x_2$. The second class also has two standard normal independent inputs, but conditioned on $4.5 \leq x_1^2 + x_2^2 \leq 8$. The dictionary of basis functions is $\mathcal{D} = \{\sqrt{2}x_1, \sqrt{2}x_2, \sqrt{2}x_1x_2, x_1^2, x_2^2\}$. The solution path $\hat{\beta}(s)$ as a function of $s$ is shown in Figure 1. Any segment between two adjacent vertical lines is linear. Hence the right derivative of $\hat{\beta}(s)$ with respect to $s$ is piece-wise constant (in $\mathcal{R}^q$). The two solid paths are for $x_1^2$ and $x_2^2$, which are the two relevant features.

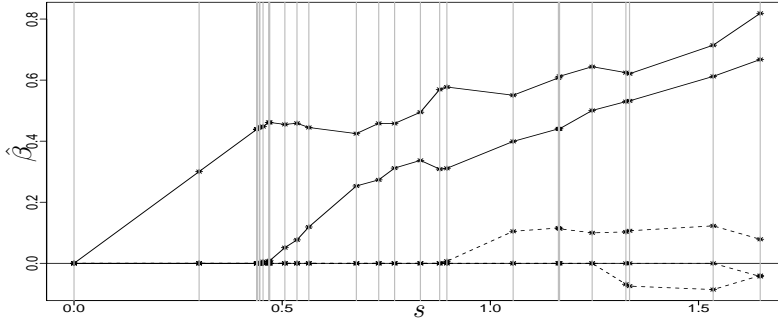

Figure 1: The solution path $\hat{\beta}(s)$ as a function of $s$.

In section 2, we motivate why we are interested in the 1-norm SVM. In section 3, we describe the algorithm that computes the whole solution path $\hat{\beta}(s)$. In section 4, we show some numerical results on both simulation data and real world data.

## 2   Regularized support vector machines

The standard 2-norm SVM is equivalent to fit a model that

$$\min_{\beta_0,\beta_j} \sum_{i=1}^{n} \left[ 1 - y_i \left( \beta_0 + \sum_{j=1}^{q} \beta_j h_j(x_i) \right) \right]_+ + \lambda \|\beta\|_2^2, \qquad (4)$$

where $\lambda$ is a tuning parameter. In practice, people usually choose $h_j(x)$'s to be the basis functions of a reproducing kernel Hilbert space. Then a kernel trick allows the dimension of the transformed feature space to be very large, even infinite in some cases (i.e. $q = \infty$), without causing extra computational burden ([2] and [12]). In this paper, however, we will concentrate on the basis representation (3) rather than a kernel representation.

Notice that (4) has the form $loss + penalty$, and $\lambda$ is the tuning parameter that controls the tradeoff between loss and penalty. The loss $(1 - yf)_+$ is called the *hinge* loss, and

the penalty is called the *ridge* penalty. The idea of penalizing by the sum-of-squares of the parameters is also used in neural networks, where it is known as *weight decay*. The ridge penalty shrinks the fitted coefficients $\hat{\beta}$ towards zero. It is well known that this shrinkage has the effect of controlling the variances of $\hat{\beta}$, hence possibly improves the fitted model's prediction accuracy, especially when there are many highly correlated features [6]. So from a statistical function estimation point of view, the ridge penalty could possibly explain the success of the SVM ([6] and [12]). On the other hand, computational learning theory has associated the good performance of the SVM to its margin maximizing property [11], a property of the hinge loss. [8] makes some effort to build a connection between these two different views.

In this paper, we replace the ridge penalty in (4) with the $L_1$-norm of $\beta$, i.e. the *lasso* penalty [10], and consider the 1-norm SVM problem:

$$\min_{\beta_0, \beta} \sum_{i=1}^{n} \left[ 1 - y_i \left( \beta_0 + \sum_{j=1}^{q} \beta_j h_j(x_i) \right) \right]_+ + \lambda \|\beta\|_1, \tag{5}$$

which is an equivalent Lagrange version of the optimization problem (1)-(2).

The lasso penalty was first proposed in [10] for regression problems, where the response $y$ is continuous rather than categorical. It has also been used in [1] and [9] for classification problems under the framework of SVMs. Similar to the ridge penalty, the lasso penalty also shrinks the fitted coefficients $\hat{\beta}$'s towards zero, hence (5) also benefits from the reduction in fitted coefficients' variances. Another property of the lasso penalty is that because of the $L_1$ nature of the penalty, making $\lambda$ sufficiently large, or equivalently $s$ sufficiently small, will cause some of the coefficients $\hat{\beta}_j$'s to be *exactly zero*. For example, when $s = 1$ in Figure 1, only three fitted coefficients are non-zero. Thus the lasso penalty does a kind of continuous feature selection, while this is not the case for the ridge penalty. In (4), none of the $\hat{\beta}_j$'s will be equal to zero.

It is interesting to note that the ridge penalty corresponds to a Gaussian prior for the $\beta_j$'s, while the lasso penalty corresponds to a double-exponential prior. The double-exponential density has heavier tails than the Gaussian density. This reflects the greater tendency of the lasso to produce some large fitted coefficients and leave others at 0, especially in high dimensional problems. Recently, [3] consider a situation where we have a small number of training data, e.g. $n = 100$, and a large number of basis functions, e.g. $q = 10,000$. [3] argue that in the *sparse* scenario, i.e. only a small number of true coefficients $\beta_j$'s are non-zero, the lasso penalty works better than the ridge penalty; while in the non-sparse scenario, e.g. the true coefficients $\beta_j$'s have a Gaussian distribution, neither the lasso penalty nor the ridge penalty will fit the coefficients well, since there is too little data from which to estimate these non-zero coefficients. This is the *curse of dimensionality* taking its toll. Based on these observations, [3] further propose the *bet on sparsity* principle for high-dimensional problems, which encourages using lasso penalty.

## 3   Algorithm

Section 2 gives the motivation why we are interested in the 1-norm SVM. To solve the 1-norm SVM for a *fixed* value of $s$, we can transform (1)-(2) into a linear programming problem and use standard software packages; but to get a good fitted model $\hat{f}(x)$ that performs well on future data, we need to select an appropriate value for the tuning paramter $s$. In this section, we propose an efficient algorithm that computes the whole solution path $\hat{\beta}(s)$, hence facilitates adaptive selection of $s$.

### 3.1 Piece-wise linearity

If we follow the solution path $\hat{\beta}(s)$ of (1)-(2) as $s$ increases, we will notice that since both $\sum_i (1 - y_i \hat{f}_i)_+$ and $\|\beta\|_1$ are piece-wise linear, the Karush-Kuhn-Tucker conditions will not change when $s$ increases unless a *residual* $(1 - y_i \hat{f}_i)$ changes from non-zero to zero, or a fitted coefficient $\hat{\beta}_j(s)$ changes from non-zero to zero, which correspond to the non-smooth points of $\sum_i (1 - y_i \hat{f}_i)_+$ and $\|\beta\|_1$. This implies that the derivative of $\hat{\beta}(s)$ with respect to $s$ is piece-wise constant, because when the Karush-Kuhn-Tucker conditions do not change, the derivative of $\hat{\beta}(s)$ will not change either. Hence it indicates that the whole solution path $\hat{\beta}(s)$ is piece-wise linear. See [13] for details.

Thus to compute the whole solution path $\hat{\beta}(s)$, all we need to do is to find the *joints*, i.e. the asterisk points in Figure 1, on this piece-wise linear path, then use straight lines to interpolate them, or equivalently, to start at $\hat{\beta}(0) = 0$, find the right derivative of $\hat{\beta}(s)$, let $s$ increase and only change the derivative when $\hat{\beta}(s)$ gets to a joint.

### 3.2 Initial solution (i.e. $s = 0$)

The following notation is used. Let $\mathcal{V} = \{j : \hat{\beta}_j(s) \neq 0\}$, $\mathcal{E} = \{i : 1 - y_i \hat{f}_i = 0\}$, $\mathcal{L} = \{i : 1 - y_i \hat{f}_i > 0\}$ and $u$ for the right derivative of $\hat{\beta}_{\mathcal{V}}(s)$: $\|u\|_1 = 1$ and $\hat{\beta}_{\mathcal{V}}(s)$ denotes the components of $\hat{\beta}(s)$ with indices in $\mathcal{V}$. Without loss of generality, we assume $\#\{y_i = 1\} \geq \#\{y_i = -1\}$; then $\hat{\beta}_0(0) = 1, \hat{\beta}_j(0) = 0$. To compute the path that $\hat{\beta}(s)$ follows, we need to compute the derivative of $\hat{\beta}(s)$ at 0. We consider a modified problem:

$$\min_{\beta_0, \beta_j} \quad \sum_{y_i=1} (1 - y_i f_i)_+ + \sum_{y_i=-1} (1 - y_i f_i) \tag{6}$$

$$\text{s.t.} \quad \|\beta\|_1 \leq \Delta s, \quad f_i = \beta_0 + \sum_{j=1}^{q} \beta_j h_j(x_i). \tag{7}$$

Notice that if $y_i = 1$, the loss is still $(1 - y_i f_i)_+$; but if $y_i = -1$, the loss becomes $(1 - y_i f_i)$. In this setup, the derivative of $\hat{\beta}(\Delta s)$ with respect to $\Delta s$ is the same no matter what value $\Delta s$ is, and one can show that it coincides with the right derivative of $\hat{\beta}(s)$ when $s$ is sufficiently small. Hence this setup helps us find the initial derivative $u$ of $\hat{\beta}(s)$. Solving (6)-(7), which can be transformed into a simple linear programming problem, we get initial $\mathcal{V}$, $\mathcal{E}$ and $\mathcal{L}$. $|\mathcal{V}|$ should be equal to $|\mathcal{E}|$. We also have:

$$\left( \begin{array}{c} \hat{\beta}_0(\Delta s) \\ \hat{\beta}_{\mathcal{V}}(\Delta s) \end{array} \right) = \left( \begin{array}{c} 1 \\ 0 \end{array} \right) + \Delta s \cdot \left( \begin{array}{c} u_0 \\ u \end{array} \right). \tag{8}$$

$\Delta s$ starts at 0 and increases.

### 3.3 Main algorithm

The main algorithm that computes the whole solution path $\hat{\beta}(s)$ proceeds as following:

1. Increase $\Delta s$ until one of the following two events happens:
   - A training point hits $\mathcal{E}$, i.e. $1 - y_i f_i \neq 0$ becomes $1 - y_i f_i = 0$ for some $i$.
   - A basis function in $\mathcal{V}$ leaves $\mathcal{V}$, i.e. $\hat{\beta}_j \neq 0$ becomes $\hat{\beta}_j = 0$ for some $j$.

   Let the current $\hat{\beta}_0$, $\hat{\beta}$ and $s$ be denoted by $\hat{\beta}_0^{old}$, $\hat{\beta}^{old}$ and $s^{old}$.

2. For each $j^* \notin \mathcal{V}$, we solve:
$$\begin{cases} u_0 + \sum_{\mathcal{V}} u_j h_j(x_i) + u_{j^*} h_{j^*}(x_i) &=& 0 \quad \text{for } i \in \mathcal{E} \\ \sum_{\mathcal{V}} sign(\hat{\beta}_j^{old}) u_j + |u_{j^*}| &=& 1 \end{cases} \tag{9}$$
where $u_0$, $u_j$ and $u_{j^*}$ are the unknowns. We then compute:
$$\frac{\Delta loss_{j^*}}{\Delta s} = \sum_{\mathcal{L}} y_i \left( u_0 + \sum_{\mathcal{V}} u_j h_j(x_i) + u_{j^*} h_{j^*}(x_i) \right). \tag{10}$$

3. For each $i' \in \mathcal{E}$, we solve:
$$\begin{cases} u_0 + \sum_{\mathcal{V}} u_j h_j(x_i) &=& 0 \quad \text{for } i \in \mathcal{E}\backslash\{i'\} \\ \sum_{\mathcal{V}} sign(\hat{\beta}_j^{old}) u_j &=& 1 \end{cases} \tag{11}$$
where $u_0$ and $u_j$ are the unknowns. We then compute:
$$\frac{\Delta loss_{i'}}{\Delta s} = \sum_{\mathcal{L}} y_i \left( u_0 + \sum_{\mathcal{V}} u_j h_j(x_i) \right). \tag{12}$$

4. Compare the computed values of $\frac{\Delta loss}{\Delta s}$ from step 2 and step 3. There are $q - |\mathcal{V}| + |\mathcal{E}| = q + 1$ such values. Choose the smallest negative $\frac{\Delta loss}{\Delta s}$. Hence,

   - If the smallest $\frac{\Delta loss}{\Delta s}$ is non-negative, the algorithm terminates; else
   - If the smallest negative $\frac{\Delta loss}{\Delta s}$ corresponds to a $j^*$ in step 2, we update
   $$\mathcal{V} \leftarrow \mathcal{V} \cup \{j^*\}, \quad u \leftarrow \begin{pmatrix} u \\ u_{j^*} \end{pmatrix}. \tag{13}$$

   - If the smallest negative $\frac{\Delta loss}{\Delta s}$ corresponds to a $i'$ in step 3, we update $u$ and
   $$\mathcal{E} \leftarrow \mathcal{E}\backslash\{i'\}, \quad \mathcal{L} \leftarrow \mathcal{L} \cup \{i'\} \text{ if necessary.} \tag{14}$$

   In either of the last two cases, $\hat{\beta}(s)$ changes as:
   $$\begin{pmatrix} \hat{\beta}_0(s^{old} + \Delta s) \\ \hat{\beta}_{\mathcal{V}}(s^{old} + \Delta s) \end{pmatrix} = \begin{pmatrix} \hat{\beta}_0^{old} \\ \hat{\beta}_{\mathcal{V}}^{old} \end{pmatrix} + \Delta s \cdot \begin{pmatrix} u_0 \\ u \end{pmatrix}, \tag{15}$$
   and we go back to step 1.

In the end, we get a path $\hat{\beta}(s)$, which is piece-wise linear.

### 3.4 Remarks

Due to the page limit, we omit the proof that this algorithm does indeed give the exact whole solution path $\hat{\beta}(s)$ of (1)-(2) (see [13] for detailed proof). Instead, we explain a little what each step of the algorithm tries to do.

Step 1 of the algorithm indicates that $\hat{\beta}(s)$ gets to a joint on the solution path and the right derivative of $\hat{\beta}(s)$ needs to be changed if either a residual $(1 - y_i \hat{f}_i)$ changes from non-zero to zero, or the coefficient of a basis function $\hat{\beta}_j(s)$ changes from non-zero to zero, when $s$ increases. Then there are two possible types of actions that the algorithm can take: (1) add a basis function into $\mathcal{V}$, or (2) remove a point from $\mathcal{E}$.

Step 2 computes the possible right derivative of $\hat{\beta}(s)$ if adding each basis function $h_{j^*}(x)$ into $\mathcal{V}$. Step 3 computes the possible right derivative of $\hat{\beta}(s)$ if removing each point $i'$ from $\mathcal{E}$. The possible right derivative of $\hat{\beta}(s)$ (determined by either (9) or (11)) is such that the training points in $\mathcal{E}$ are kept in $\mathcal{E}$ when $s$ increases, until the next joint (step 1) occurs. $\Delta loss/\Delta s$ indicates how fast the $loss$ will decrease if $\hat{\beta}(s)$ changes according to $u$. Step 4 takes the action corresponding to the smallest negative $\Delta loss/\Delta s$. When the $loss$ can not be decreased, the algorithm terminates.

Table 1: Simulation results of 1-norm and 2-norm SVM

| | | Test Error (SE) | | | | |
|---|---|---|---|---|---|---|
| | **Simulation** | 1-**norm** | 2-**norm** | **No Penalty** | $|\mathcal{D}|$ | **# Joints** |
| 1 | No noise input | 0.073 (0.010) | 0.08 (0.02) | 0.08 (0.01) | 5 | 94 (13) |
| 2 | 2 noise inputs | 0.074 (0.014) | 0.10 (0.02) | 0.12 (0.03) | 14 | 149 (20) |
| 3 | 4 noise inputs | 0.074 (0.009) | 0.13 (0.03) | 0.20 (0.05) | 27 | 225 (30) |
| 4 | 6 noise inputs | 0.082 (0.009) | 0.15 (0.03) | 0.22 (0.06) | 44 | 374 (52) |
| 5 | 8 noise inputs | 0.084 (0.011) | 0.18 (0.03) | 0.22 (0.06) | 65 | 499 (67) |

### 3.5 Computational cost

We have proposed an algorithm that computes the whole solution path $\hat{\beta}(s)$. A natural question is then what is the computational cost of this algorithm? Suppose $|\mathcal{E}| = m$ at a joint on the piece-wise linear solution path, then it takes $O(qm^2)$ to compute step 2 and step 3 of the algorithm through Sherman-Morrison updating formula. If we assume the training data are separable by the dictionary $\mathcal{D}$, then all the training data are eventually going to have loss $(1 - y_i \hat{f}_i)_+$ equal to zero. Hence it is reasonable to assume the number of joints on the piece-wise linear solution path is $O(n)$. Since the maximum value of $m$ is $\min(n, q)$ and the minimum value of $m$ is 1, we get the worst computational cost is $O(nq \min(n, q)^2)$ and the best computational cost is $O(nq)$. Notice that this is a rough calculation of the computational cost under some mild assumptions. Simulation results (section 4) actually indicate that the number of joints tends to be $O(\min(n, q))$.

## 4 Numerical results

In this section, we use both simulation and real data results to illustrate the 1-norm SVM.

### 4.1 Simulation results

The data generation mechanism is the same as the one described in section 1, except that we generate 50 training data in each of two classes, and to make harder problems, we sequentially augment the inputs with additional two, four, six and eight standard normal noise inputs. Hence the second class almost completely surrounds the first, like the skin surrounding the oragne, in a two-dimensional subspace. The Bayes error rate for this problem is 0.0435, irrespective of dimension. In the original input space, a hyperplane cannot separate the classes; we use an enlarged feature space corresponding to the 2nd degree polynomial kernel, hence the dictionary of basis functions is $\mathcal{D} = \{\sqrt{2}x_j, \sqrt{2}x_j x_{j'}, x_j^2, j, j' = 1, \ldots p\}$. We generate 1000 test data to compare the 1-norm SVM and the standard 2-norm SVM. The average test errors over 50 simulations, with different numbers of noise inputs, are shown in Table 1. For both the 1-norm SVM and the 2-norm SVM, we choose the tuning parameters to minimize the test error, to be as fair as possible to each method. For comparison, we also include the results for the non-penalized SVM.

From Table 1 we can see that the non-penalized SVM performs significantly worse than the penalized ones; the 1-norm SVM and the 2-norm SVM perform similarly when there is no noise input (line 1), but the 2-norm SVM is adversely affected by noise inputs (line 2 - line 5). Since the 1-norm SVM has the ability to select relevant features and ignore redundant features, it does not suffer from the noise inputs as much as the 2-norm SVM. Table 1 also shows the number of basis functions $q$ and the number of joints on the piece-wise linear solution path. Notice that $q < n$ and there is a striking linear relationship between $|\mathcal{D}|$ and $\#Joints$ (Figure 2). Figure 2 also shows the 1-norm SVM result for one simulation.

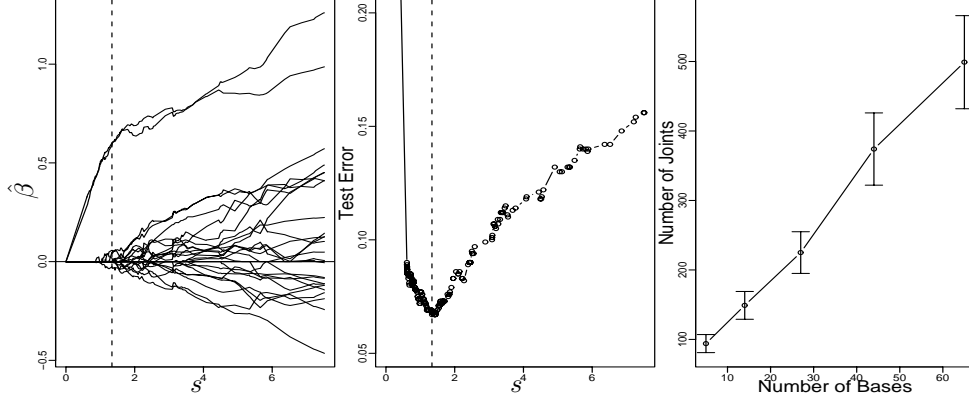

Figure 2: Left and middle panels: 1-norm SVM when there are 4 noise inputs. The left panel is the piece-wise linear solution path $\hat{\beta}(s)$. The two upper paths correspond to $x_1^2$ and $x_2^2$, which are the relevant features. The middle panel is the test error along the solution path. The dash lines correspond to the minimum of the test error. The right panel illustrates the linear relationship between the number of basis functions and the number of joints on the solution path when $q < n$.

## 4.2    Real data results

In this section, we apply the 1-norm SVM to classification of gene microarrays. Classification of patient samples is an important aspect of cancer diagnosis and treatment. The 2-norm SVM has been successfully applied to microarray cancer diagnosis problems ([5] and [7]). However, one weakness of the 2-norm SVM is that it only predicts a cancer class label but does not automatically select relevant genes for the classification. Often a primary goal in microarray cancer diagnosis is to identify the genes responsible for the classification, rather than class prediction. [4] and [5] have proposed gene selection methods, which we call univariate ranking (UR) and recursive feature elimination (RFE) (see [14]), that can be combined with the 2-norm SVM. However, these procedures are two-step procedures that depend on external gene selection methods. On the other hand, the 1-norm SVM has an inherent gene (feature) selection property due to the lasso penalty. Hence the 1-norm SVM achieves the goals of classification of patients and selection of genes simultaneously.

We apply the 1-norm SVM to leukemia data [4]. This data set consists of 38 training data and 34 test data of two types of acute leukemia, *acute myeloid leukemia* (AML) and *acute lymphoblastic leukemia* (ALL). Each datum is a vector of $p = 7,129$ genes. We use the original input $x_j$, i.e. the $j$th gene's expression level, as the basis function, i.e. $q = p$. The tuning parameter is chosen according to 10-fold cross-validation, then the final model is fitted on all the training data and evaluated on the test data. The number of joints on the solution path is $104$, which appears to be $O(n) \ll O(q)$. The results are summarized in Table 2. We can see that the 1-norm SVM performs similarly to the other methods in classification and it has the advantage of automatically selecting relevant genes. We should notice that the maximum number of genes that the 1-norm SVM can select is upper bounded by $n$, which is usually much less than $q$ in microarray problems.

## 5    Conclusion

We have considered the 1-norm SVM in this paper. We illustrate that the 1-norm SVM may have some advantage over the 2-norm SVM, especially when there are redundant features. The solution path $\hat{\beta}(s)$ of the 1-norm SVM is a piece-wise linear function in the tuning

Table 2: Results on Microarray Classification

| Method | CV Error | Test Error | # of Genes |
|---|---|---|---|
| 2-norm SVM UR | 2/38 | 3/34 | 22 |
| 2-norm SVM RFE | 2/38 | 1/34 | 31 |
| 1-norm SVM | 2/38 | 2/34 | 17 |

parameter $s$. We have proposed an efficient algorithm to compute the whole solution path $\hat{\beta}(s)$ of the 1-norm SVM, and facilitate adaptive selection of the tuning parameter $s$.

## Acknowledgments

Hastie was partially supported by NSF grant DMS-0204162, and NIH grant ROI-CA-72028-01. Tibshirani was partially supported by NSF grant DMS-9971405, and NIH grant ROI-CA-72028.

## References

[1] Bradley, P. & Mangasarian, O. (1998) Feature selection via concave minimization and support vector machines. In J. Shavlik (eds), *ICML'98*. Morgan Kaufmann.

[2] Evgeniou, T., Pontil, M. & Poggio., T. (1999) Regularization networks and support vector machines. *Advances in Large Margin Classifiers*. MIT Press.

[3] Friedman, J., Hastie, T, Rosset, S, Tibshirani, R. & Zhu, J. (2004) Discussion of "Consistency in boosting" by W. Jiang, G. Lugosi, N. Vayatis and T. Zhang. *Annals of Statistics*. To appear.

[4] Golub,T., Slonim,D., Tamayo,P., Huard,C., Gaasenbeek,M., Mesirov,J., Coller,H., Loh,M., Downing,J. & Caligiuri,M. (1999) Molecular classification of cancer: class discovery and class prediction by gene expression monitoring. *Science* **286**, 531-536.

[5] Guyon,I., Weston,J., Barnhill,S. & Vapnik,V. (2002) Gene selection for cancer classification using support vector machines. *Machine Learning* **46**, 389-422.

[6] Hastie, T., Tibshirani, R. & Friedman, J. (2001) *The Elements of Statistical Learning*. Springer-Verlag, New York.

[7] Mukherjee, S., Tamayo,P., Slonim,D., Verri,A., Golub,T., Mesirov,J. & Poggio, T. (1999) Support vector machine classification of microarray data. *Technical Report AI Memo 1677, MIT*.

[8] Rosset, S., Zhu, J. & Hastie, T. (2003) Boosting as a regularized path to a maximum margin classifer. *Technical Report, Department of Statistics, Stanford University, CA*.

[9] Song, M., Breneman, C., Bi, J., Sukumar, N., Bennett, K., Cramer, S. & Tugcu, N. (2002) Prediction of protein retention times in anion-exchange chromatography systems using support vector regression. *Journal of Chemical Information and Computer Sciences*, September.

[10] Tibshirani, R. (1996) Regression shrinkage and selection via the lasso. *J.R.S.S.B.* **58**, 267-288.

[11] Vapnik, V. (1995) *Tha Nature of Statistical Learning Theory*. Springer-Verlag, New York.

[12] Wahba, G. (1999) Support vector machine, reproducing kernel Hilbert spaces and the randomized GACV. *Advances in Kernel Methods - Support Vector Learning*, 69-88, MIT Press.

[13] Zhu, J. (2003) Flexible statistical modeling. *Ph.D. Thesis*. Stanford University.

[14] Zhu, J. & Hastie, T. (2003) Classification of gene microarrays by penalized logistic regression. *Biostatistics*. Accepted.